# Active Learning Applied to Patient-Adaptive Heartbeat Classification

**Jenna Wiens**
CSAIL, MIT
jwiens@csail.mit.edu

**John V. Guttag**
CSAIL, MIT
guttag@csail.mit.edu

## Abstract

While clinicians can accurately identify different types of heartbeats in electrocardiograms (ECGs) from different patients, researchers have had limited success in applying supervised machine learning to the same task. The problem is made challenging by the variety of tasks, inter- and intra-patient differences, an often severe class imbalance, and the high cost of getting cardiologists to label data for individual patients. We address these difficulties using active learning to perform patient-adaptive and task-adaptive heartbeat classification. When tested on a benchmark database of cardiologist annotated ECG recordings, our method had considerably better performance than other recently proposed methods on the two primary classification tasks recommended by the Association for the Advancement of Medical Instrumentation. Additionally, our method required over 90% less patient-specific training data than the methods to which we compared it.

## 1 Introduction

In 24 hours an electrocardiogram (ECG) can record over 100,000 heartbeats for a single patient. Of course, a physician is not likely to look at all of them. Automated analysis of long-term ECG recordings can help physicians understand a patient's physiological state and his/her risk for adverse cardiovascular outcomes [1] [2]. Often, an important step in such analysis is labeling the different types of heartbeats. This labeling reduces an ECG to a set of symbols transferable across patients.

Trained clinicians can successfully identify over a dozen different types of heartbeats in ECG recordings. However, researchers have had limited success using supervised machine learning techniques to do the same. The problem is made challenging by the inter-patient differences present in the morphology and timing characteristics of the ECGs produced by compromised cardiovascular systems. The variation in the physiological systems that produce the data means that a classifier trained on even a large set of patients will yield unpredictable results when applied to a new cardiac patient. For this reason, *global* classifiers are highly unreliable and therefore not widely used in practice [3].

Hu *et al* was one of the first to describe an automatic *patient-adaptive* ECG beat classifier [4]. It distinguished ventricular ectopic beats (VEBs), from non-VEBs. This work employed a mixture of experts approach, combining a global classifier with a local classifier trained on the first 5 minutes of the test patient's record. Similarly, de Chazal *et al* augmented the performance of a global heartbeat classifier by including patient-specific expert knowledge for each test patient. Their local classifier was trained on the first 500 labeled beats of each record [3]. More recently, Ince *et al* developed a patient-adaptive classification scheme using artificial neural networks by incorporating the first 5 minutes of each test recording in the training set [5].

Based on the results from these three studies, it is clear that patient-adaptive classifiers provide increased classification accuracy. Unfortunately, patient-adaptive classifiers are not used in practice because they require an unrealistic amount of labor to produce a cardiologist-labeled patient-specific training set. Furthermore, by sampling all of the patient-specific training data from one portion of

the ECG, one is at risk for over-fitting to that patient's physiological state in time. Given a long-term record, which is likely to contain high intra-patient differences, it is likely that constructing the training set in this manner will not yield a good representation of the patient's ECG.

There has been some success with hand-coded rule-based algorithms for heartbeat classification. Hamilton *et al* developed a rule-based algorithm for detecting one type of particularly dangerous ectopic heartbeat, the premature ventricular contraction (PVC) [6]. While reasonably accurate, rule-based algorithms are inflexible, since they can only be used for a single classification task. And to be useful in practice, a classifier should not only be capable of adapting to new patients, but also to new classification problems, since the classification task in question can change depending on the patient or even the clinician. Since the field of ECG research is continuously evolving, tools to analyze the signal should be capable of adapting.

In this paper, we show how active learning can be successfully applied to the problems of both patient-adaptive and task-adaptive heartbeat classification. We developed our method with a clinical setting in mind: initially it requires no labeled data, it has no user-specified parameters, and achieves good performance on an imbalanced data set. Applied to data from the MIT-BIH Arrhythmia Database our method outperforms current state-of-the-art machine learning heartbeat classification techniques and uses less training data. Moreover, our approach outperforms a rule-based algorithm designed to detect an important class of abnormal beat. Finally, we discuss how the classification method performed when used in a prospective experiment with two cardiologists.

## 2 Background

We begin with a brief background on the signal of interest, the ECG. Since we will consider different heartbeat classification tasks we first present a few examples of heartbeat classes and ECG abnormalities.

### 2.1 The ECG and ECG Abnormalities

An ECG records a patient's cardiac electrical activity by measuring the potential differences at the surface of the patient's body. In most healthy patients, the ECG, measured from Lead II, begins with a P-wave, is followed by a QRS complex and ends with a T-wave. Figure 1(a) shows an example of the ECG of a normal sinus rhythm beat (N). The exact morphology and timing of the different portions of the wave depend on the patient and lead placement.

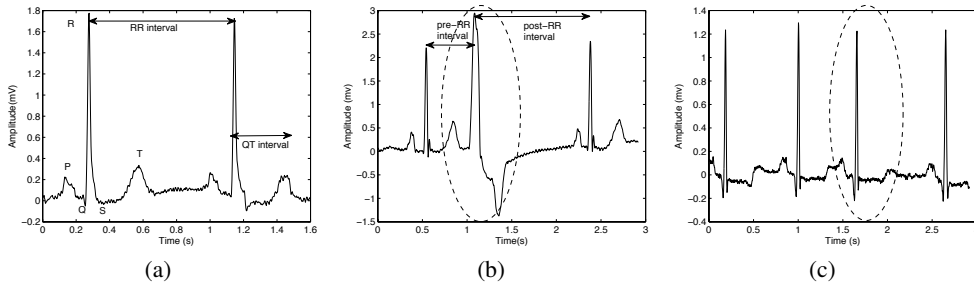

|(a)|(b)|(c)|

Figure 1: Normal sinus rhythm beats like the ones shown in (a) originate from the pacemaker cells of the sinoatrial node. Premature ventricular contractions (b) and atrial premature beats (c) are two examples of ectopic beats.

Cardiac abnormalities can disrupt the heart's normal sinus rhythm, and, depending on their type and frequency, can vary from benign to life threatening. Examples of ectopic beats (beats that do not originate in the sinoatrial node) are shown in Figures 1(b) and 1(c). Premature ventricular contractions (PVCs), originate in the ventricles instead of in the pacemaker cells of the sinoatrial node. They are common in patients who have suffered an acute myocardial infarction [7] and may indicate that a patient is at increased risk for more serious ventricular arrhythmias and sudden cardiac death [8]. When the electrical impulse originates from the atria, an atrial premature beat is recorded by the ECG as shown in Figure 1(c). Atrial premature beats tend not to be life threatening.

Because of their specific timing and morphology characteristics these two types of abnormal beats are generally distinguishable by trained cardiologists, but there are many exceptions. Not only can abnormalities vary from patient to patient, but the same recording may contain beats that belong to the same class but all look quite different. Figure 2 shows an example of an ECG containing multiform PVCs.

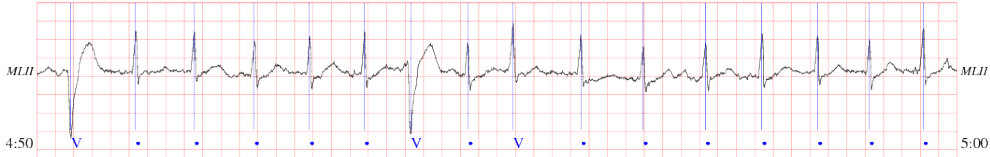

Figure 2: Each PVC is marked by a "V" and each normal sinus rhythm beat is marked by a "·". The PVC morphology varies greatly among patients and even within recordings from a single patient.

## 3   Methods

In this section we describe the two main components of our heartbeat classification scheme. We begin, with the process of feature extraction and then present the classification method.

### 3.1   Feature Extraction

Before extracting feature vectors, we pre-process and segment the ECG. We used PhysioNet's automated R-peak detector to detect the R-peaks of each heartbeat [9]. Next, we removed baseline wander from the signals using the method described in [10]. Once pre-processed, the data was segmented into individual heartbeats based on fixed intervals before and after the R-peak, so that each beat contained the same number of samples.

Our goal was to develop a feature vector that worked well not only across patients but also across different heartbeat classification tasks. This led us to use a combination of the ECG features proposed in [10],[11], and [12]. The elements of the feature vector, $\underline{x}$, are described in Table 1.

Table 1: Heartbeat features used in experiments.

| Features | Description |
|---|---|
| $x_1, ..., x_{60}$ | • Wavelet coefficients from the last 5 levels of a 6 level wavelet decomposition using a Daubechies 2 wavelet |
| $x_{61}, x_{62}, x_{63}$ | • The normalized energy in different segments of the beat |
| $x_{64}, x_{65}, x_{66}$ | • The pre and post RR intervals normalized by a local average, and the average RR interval |
| $x_{67}$ | • Morphological distance between the current beat the record's median beat |

The last, and most novel, feature in Table 1 is a measure of the morphological distance between the represented beat and the median beat for a patient (recalculated every 500 beats). The feature is based on the dynamic time warping algorithm used in [12] to measure the morphological distance between a fixed interval that contains a portion of the Q-T intervals of two beats.

### 3.2   Classification

Our goal was to develop a clinically useful patient-adaptive heartbeat classification method for solving different binary heartbeat classification problems. We designed the classifier for use in a clinical setting, where physicians have little time to label beats, let alone tune classifier parameters. Thus, it was important that the method should require few cardiologist-labeled heartbeats, and have no user-defined parameters. Based on these goals we developed the algorithm presented below, which combines different ideas from the literature [13-16].

Inputs:

(a) Unlabeled data $\{x_1, ..., x_n\}$

(b) Max number of initial clusters per clustering, $k$

(c) SVM cost parameter $C$

(d) Stopping precision $\epsilon$

1. Cluster the data using hierarchical clustering with two different linkage criteria, yielding $<= 2 * k$ clusters.

2. Query the centroid of each cluster. Add these points to the initially empty set of labeled examples.

3. If the expert labeled all the points as belonging to the same class, stop, else $k = 1$.

4. Train a linear SVM based on the labeled examples.

5. Apply the SVM to all of the data.

6. If all data that lies on or within the margin is labeled, stop.

7. Re-cluster data that lie on or within the margin using hierarchical clustering with $k = k + 1$.

8. Query the point from each cluster that lies closest to the current SVM decision boundary.

9. Repeat steps 4-8 until the change in the margin is within $\epsilon$ of zero.

Many proposed techniques for SVM active learning assume one starts with some set of labeled data or, as in [13], the initial training examples are randomly selected. In our application, we start with a pool of completely unlabeled data. Furthermore, since there is often a severe class imbalance (*e.g.*, some multi-thousand beat recordings contain less than a handful of PVCs), choosing a small or even moderate number of random samples is unlikely to be an effective approach to finding representative samples of a record. The choice of initial queries is crucial. If beats from only one class are queried the algorithm could stop prematurely. More generally, the selection of the first set of queries is independent of the binary task, and therefore the first query should contain at least one example from each of the beat classes contained in the record. We use clustering in an effort to quickly identify representative samples from each class.

We experimented with different clustering techniques before choosing hierarchical clustering. On average hierarchical clustering outperformed other popular clustering techniques like $k$-means. We believe this can be attributed to the fact that hierarchical clustering has the ability to produce a variety of different clusters by modifying the linkage criterion. We chose to use two complementary linkage criteria in attempt to address the intra-patient variation present in ECG records. The first metric is average linkage. Average linkage defines the distance between two clusters, $q$ and $r$, as the average distance between all pairs of objects in $q$ and $r$. This linkage is biased toward producing clusters with similar variances, and has the tendency to merge clusters with small variances. The second linkage criterion is Ward's linkage [17], defined in Equation 1.

$$d(q, r) = ss(qr) - [ss(q) + ss(r)] \tag{1}$$

where $ss(qr)$ is the within-cluster sum of squares for the resulting cluster when $q$ and $r$ are combined. The within-cluster sum of squares, $ss(x)$, is defined as the sum of squares of the distances between all objects in the cluster and the centroid of the cluster:

$$ss(x) = \sum_{i=1}^{n_x} |x_i - \frac{1}{n_x} \sum_{j=1}^{n_x} x_j|^2 \tag{2}$$

Using Ward's linkage tends to join clusters with a small number of points, and is biased towards producing clusters with approximately the same number of samples. If presented with an outlier, Ward's method tends to assign it to the cluster with the closest centroid, whereas the average linkage tends to assign it to the densest cluster, where it will have the smallest impact on the maximum variance [18].

Once the initial queries are labeled, we train a linear SVM, and apply this SVM to all of the data. We use linear SVMs because most heartbeat classification tasks are close to linearly separable and because linear SVMs require few tuning parameters. Next, we re-cluster the data on or within the margin of the SVM, incrementing the max number of clusters with each iteration. We then query a beat from each cluster that is closest to the SVM decision boundary.

As described above, our algorithm would halt when no unlabeled data lay on or within the margin. For some records, however, *e.g.*, those with *fusion* beats - a fusion of normal and abnormal beats

- many beats can lie within the margin of the SVM and thus a clinician might end up labeling hundreds of beats that add little useful information. Intuitively, one should stop querying when additional training data has little to no effect on the solution. The algorithm, therefore, terminates when the change in the margin between iterations is within $\epsilon$.

## 4  Experiments & Results

We implemented our algorithm in MATLAB, and used $SVM_{light}$ [19] to train the linear SVM at each iteration. We held the cost parameter of the linear SVM constant, at $C = 100$, throughout all experiments. This value was selected based on previous cross-validation experiments. The stopping precision $\epsilon$ was held constant at $\epsilon = 10^{-3}$. Typical ECG recordings contain beats from 2 to 5 classes but can contain more; based on this *a priori* knowledge, we conservatively set $k = 10$. This value was held constant throughout all experiments.

To test the utility of our proposed approach for heartbeat classification we ran a series of experiments on data from different patients, and for different classification tasks. First, we compare the performance of a classifier obtained using our approach to two classifiers recently presented in the literature. Next, we directly measure the impact active learning has on the classification of heartbeats by creating our own passive learning classifier using the same pre-processing and features as our proposed active learning method. Finally, we test our method using actual cardiologists.

In our experiments we report the classification performance in terms of sensitivity (SE), specificity (SP), and positive predictive value (PPV). As an overall measure of performance we use the F-score:

$$F = \frac{2 * SE * PPV}{SE + PPV} \tag{3}$$

The F-score is a commonly-accepted performance evaluation measure in medicine and information retrieval where one data class (often the positive class) is more important than the other [20]. We use this measure since the problem of heartbeat classification suffers from severe class imbalance, and thus the SE (aka recall) and the PPV (aka precision) are more important than SP.

### 4.1  Classification Performance

We tested performance on the MIT-BIH Arrhythmia Database (MITDB) [9], a widely used benchmark database that contains 48 half-hour ECG recordings, sampled at 360Hz, from 47 different patients. Twenty-three of these records, labeled 100 to 124 were selected at random from a source of 4000 recordings. The remaining 25 records, labeled 200 to 234 were selected because they contain rare clinical activity that might not have been represented had all 48 records been chosen at random. The database contains approximately 109,000 cardiologist labeled heartbeats. Each beat is labeled as belonging to one of 16 different classes. In some sense, the data in the MITDB is too good. It was collected at 360Hz, which is a higher sampling rate than is typical for the Holter monitors used to gather most long term clinical data. To simulate this kind of data, we resampled the pre-processed ECG signal at 128Hz.

We consider the two main classification tasks proposed by the Association for the Advancement of Medical Instrumentation (AAMI): detecting ventricular ectopic beats (VEBs), and detecting supraventricular ectopic beats (SVEBs). These two tasks have been the focus of other researchers investigating patient-adaptive heartbeat classification. Recently, Ince *et al* [5] and de Chazal *et al* [3] described methods that combine global information with patient-specific information. Ince *et al* trained a global classifier on 245 hand chosen beats from the MITDB, and then adapted the global classifier by training on labeled data from the first five minutes of each test record. Their reported results of testing on 44 of the 48 records - all records with paced beats were excluded - from the MITDB are reported in Table 2. De Chazal *et al* trained their global classifier on all of the data from 22 patients in the MITDB, and then adapted the global classifier by training on labeled data for the first 500 beats of each test record. Their reported results of testing on 22 records -different from the ones used in the global training set- from the MITDB are also reported in Table 2.

For the same two classification tasks we tested our proposed approach and we report the results when tested on the records reported on in [5] and [3]. In these experiments we exclude the queried

beats from the test set, testing only on data the expert hasn't seen. This was also done in [5] and [3]. Since we query far fewer beats that the other methods, we end up testing on many more beats.

Table 2: Our proposed method outperforms other classifiers for two common classification tasks.

| Classifier | VEB | | | | SVEB | | | |
|---|---|---|---|---|---|---|---|---|
| | SE | SP | PPV | **F-Score** | Sens | Spec | PPV | **F-Score** |
| Ince et al | 84.6% | 98.7% | 87.4% | **86.0%** | 63.5% | 99.0% | 53.7% | **58.2%** |
| Proposed[1] | 99.0% | 99.9% | 99.2% | **99.1%** | 88.3% | 100.0% | 99.2% | **93.4%** |
| Chazal et al | 94.3% | 99.7% | 96.2% | **95.2%** | 87.7% | 96.2% | 47.0% | **61.2%** |
| Proposed[2] | 99.6% | 99.9% | 99.3% | **99.5%** | 92.0% | 100.0% | 99.5% | **95.6%** |

[1] for the 44 records in common
[2] for the 22 records in common

As Table 2 shows, the method proposed here does considerably better than the methods proposed in [5] and [3] for each task. For the task of classifying VEBs vs. non-VEBs, our method on average used 45 labeled beats (compared to roughly 350 beats for [5] and 500 beats for [3]) per record. For the task of detecting SVEBs, our method used even fewer labeled beats. Recognizing SVEBs is considerably more difficult than detecting VEBs since the class imbalance problem is even more severe and supra-ventricular beats are harder to distinguish from normal sinus rhythm beats.

Table 3: Our algorithm outperforms a rule-based classifier designed specifically for the task of detecting PVCs.

| Classifier | SE | SP | PPV | **F-Score** |
|---|---|---|---|---|
| Hamilton *et al* | 92.8% | 98.4% | 79.5% | **85.7%** |
| Proposed[3] | 99.0% | 100.0% | 99.3% | **99.1%** |

[3] for all 48 records

Hamilton *et al* proposed a rule-based classifier for classifying PVCs vs. non-PVCs. Their software is freely available online, from *eplimited.com*. We applied their software to all of the records, see Table 3. Their method does particularly poorly on the four records containing paced beats. Omitting these four records the F-Score increases to 91.4%, still worse than our method. One advantage of the rule-based algorithm is that it does not require a labeled training set, whereas on average we require 45 labeled beats per record. However, unlike our method the rule-based algorithm can only be used for one task.

## 4.2 The Impact of Active Learning

We hypothesize that the difference in performance between our method and the other learning-based methods discussed above is attributable partly to the design of our feature vector and partly to the method of choosing training data. In order to test this hypothesis we ran an experiment that directly compares the effect of actively vs. passively selecting the training set, with all other parameters kept the same (e.g., identical pre-processing, identical feature vectors, etc.).

For each of the 48 records in the MITDB we compare a VEB vs. non-VEB classifier using our approach, to a linear SVM classifier trained on the first 500 beats of each record. For each patient we record the number of queries made, as well as the performance of each classifier. Table 4 shows the classification results for each method across all patients. The column headed "#Q" gives the number of beats used for training each classifier, while the column headed "TP" for true positives, gives the number of correctly labeled VEBs. The last row gives the totals across all records for each classification method.

Overall, our classification approach achieves an F-score over 99%, and the passive technique achieves an F-score of 94%. Compared to the passive approach, active learning used over 90% less training data, and resulted in over 85% fewer misclassified heartbeats. These results emphasize that fact that active learning can be used to dramatically reduce the labor cost of producing highly accurate classifiers. That the passive technique performed better than [5] and almost as well as [3], despite not having any global training data, suggests that our feature vector provides some advantage.

Table 4: Active versus passive learning. Active learning outperforms a passive approach, and uses over 90% less data.

| | Proposed | | | | | Passive | | | | |
|---|---|---|---|---|---|---|---|---|---|---|
| | # Q | TP | TN | FP | FN | # Q | TP | TN | FP | FN |
| 100 | 22 | 1 | 2258 | 0 | 0 | 500 | 0 | 2269 | 0 | 1 |
| 101 | 19 | 0 | 1851 | 0 | 0 | 500 | 0 | 1861 | 0 | 0 |
| 102 | 28 | 4 | 2162 | 0 | 0 | 500 | 3 | 2181 | 0 | 1 |
| 103 | 20 | 0 | 2073 | 0 | 0 | 500 | 0 | 2082 | 0 | 0 |
| 104 | 30 | 2 | 2214 | 0 | 0 | 500 | 1 | 2224 | 0 | 1 |
| 105 | 54 | 41 | 2501 | 0 | 0 | 500 | 41 | 2521 | 1 | 0 |
| 106 | 50 | 520 | 1497 | 0 | 0 | 500 | 507 | 1506 | 0 | 13 |
| 107 | 31 | 59 | 2070 | 0 | 0 | 500 | 11 | 2076 | 0 | 48 |
| 108 | 52 | 17 | 1717 | 0 | 0 | 500 | 17 | 1740 | 4 | 0 |
| 109 | 45 | 38 | 2463 | 0 | 0 | 500 | 36 | 2492 | 0 | 2 |
| 111 | 22 | 1 | 2107 | 0 | 0 | 500 | 0 | 2122 | 0 | 1 |
| 112 | 20 | 0 | 2529 | 0 | 0 | 500 | 0 | 2537 | 0 | 0 |
| 113 | 19 | 0 | 1783 | 0 | 0 | 500 | 0 | 1793 | 0 | 0 |
| 114 | 51 | 43 | 1807 | 0 | 0 | 500 | 42 | 1833 | 2 | 1 |
| 115 | 20 | 0 | 1942 | 0 | 0 | 500 | 0 | 1951 | 0 | 0 |
| 116 | 34 | 109 | 2283 | 0 | 0 | 500 | 106 | 2301 | 0 | 3 |
| 117 | 20 | 0 | 1523 | 0 | 0 | 500 | 0 | 1533 | 0 | 0 |
| 118 | 30 | 16 | 2242 | 0 | 0 | 500 | 4 | 2260 | 0 | 12 |
| 119 | 32 | 444 | 1523 | 0 | 0 | 500 | 444 | 1542 | 0 | 0 |
| 121 | 24 | 1 | 1849 | 0 | 0 | 500 | 0 | 1860 | 0 | 1 |
| 122 | 20 | 0 | 2464 | 0 | 0 | 500 | 0 | 2473 | 0 | 0 |
| 123 | 26 | 3 | 1500 | 0 | 0 | 500 | 3 | 1513 | 0 | 0 |
| 124 | 32 | 41 | 1558 | 0 | 6 | 500 | 30 | 1570 | 0 | 17 |
| 200 | 124 | 825 | 1717 | 0 | 1 | 500 | 799 | 1773 | 0 | 27 |
| 201 | 45 | 198 | 1737 | 0 | 0 | 500 | 0 | 1764 | 0 | 198 |
| 202 | 41 | 19 | 2088 | 0 | 0 | 500 | 19 | 2114 | 1 | 0 |
| 203 | 103 | 410 | 2456 | 15 | 34 | 500 | 397 | 2453 | 79 | 47 |
| 205 | 36 | 70 | 2574 | 0 | 1 | 500 | 65 | 2583 | 0 | 6 |
| 207 | 109 | 203 | 2016 | 4 | 7 | 500 | 190 | 2060 | 59 | 20 |
| 208 | 90 | 986 | 1916 | 0 | 6 | 500 | 977 | 1953 | 5 | 15 |
| 209 | 29 | 1 | 2993 | 0 | 0 | 500 | 0 | 3002 | 0 | 1 |
| 210 | 90 | 190 | 2392 | 1 | 5 | 500 | 180 | 2434 | 16 | 15 |
| 212 | 20 | 0 | 2736 | 0 | 0 | 500 | 0 | 2746 | 0 | 0 |
| 213 | 137 | 215 | 2985 | 20 | 5 | 500 | 157 | 3016 | 12 | 63 |
| 214 | 53 | 256 | 1988 | 0 | 0 | 500 | 254 | 2002 | 1 | 2 |
| 215 | 52 | 164 | 3168 | 0 | 0 | 500 | 164 | 3196 | 1 | 0 |
| 217 | 61 | 162 | 2009 | 0 | 0 | 500 | 159 | 2045 | 0 | 3 |
| 219 | 41 | 63 | 2065 | 0 | 1 | 500 | 52 | 2089 | 0 | 12 |
| 220 | 20 | 0 | 2035 | 0 | 0 | 500 | 0 | 2045 | 0 | 0 |
| 221 | 33 | 396 | 2022 | 0 | 0 | 500 | 393 | 2030 | 0 | 3 |
| 222 | 20 | 0 | 2472 | 0 | 0 | 500 | 0 | 2480 | 0 | 0 |
| 223 | 86 | 473 | 2094 | 7 | 0 | 500 | 321 | 2119 | 11 | 152 |
| 228 | 66 | 362 | 1662 | 0 | 0 | 500 | 356 | 1690 | 0 | 6 |
| 230 | 30 | 1 | 2242 | 0 | 0 | 500 | 0 | 2253 | 0 | 1 |
| 231 | 24 | 2 | 1552 | 0 | 0 | 500 | 2 | 1567 | 0 | 0 |
| 232 | 20 | 0 | 1771 | 0 | 0 | 500 | 0 | 1779 | 0 | 0 |
| 233 | 91 | 830 | 2216 | 0 | 0 | 500 | 810 | 2245 | 1 | 20 |
| 234 | 26 | 3 | 2731 | 0 | 0 | 500 | 0 | 2749 | 0 | 3 |
| **Totals** | **2148** | **7169** | **102573** | **47** | **66** | **24000** | **6540** | **102427** | **193** | **695** |

## 4.3 Experiments with Clinicians

To get a sense of the feasibility of using our approach in an actual clinical setting, we ran an experiment with two cardiologists and data from another cohort of patients admitted with NSTEACS. The ECG tracings in this database, unlike those in the MITDB, are not particularly clean, i.e., they contain a considerable amount of noise and many artifacts. This makes them more representative of the data with which an algorithm in clinical use is likely to have to deal. We considered 4 randomly chosen records, from a subset of patients who had experienced at least one episode of ventricular tachycardia in the 7 day period following randomization. For each record, we consider the first half-hour, giving us a test set of 8230 heartbeats.

In these experiments we used a slightly different stopping criterion developed earlier. As our algorithm chose beats to be labeled, each cardiologist was presented with an ECG plot of the heartbeat to be labeled and the beats surrounding it, like the one shown in Figure 3. The cardiologist was then asked to label it according to the following key: 1=clearly non-PVC , 2 = ambiguous non-PVC, 3=ambiguous PVC, 4=clearly PVC. Because the cardiologists made different choices about how some beats should be labeled, one was asked to label an average of 15 beats/record and the other roughly 20 beats/record. The whole process took each cardiologist about 90 seconds per record.

Since the records had not been previously labeled (and it seemed unreasonable to ask our experts to label all of them), we used the PVC classification software from [6] to provide a label to which

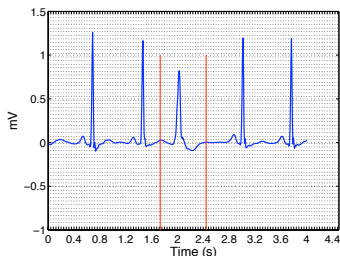

Figure 3: The classifiers trained using active learning both labeled the delineated beat delineated as a PVC, whereas the rule-based algorithm labeled it as a non-PVC.

Table 5: Comparison of active earning using two different experts and Hamilton et al. Results are the sum across four records.

| All Records (8230 beats total) | | | | | |
|---|---|---|---|---|---|
| Classifier | Size Training Data | TP | TN | FP | FN |
| Expert #1 | 60 | 191 | 8038 | 0 | 1 |
| Expert #2 | 83 | 192 | 8035 | 3 | 0 |
| Hamilton *et al* | 0 | 190 | 8035 | 3 | 2 |

we could compare the labels generated by our method. This gave us three independently generated labels for each beat. When all three classifiers agreed, we assumed that the beat was correctly classified. Out of a possible 8230 disagreements there were only 6. We asked a third expert to adjudicate all 6 disagreements, and used this as the gold standard to calculate the results for the three classifiers shown in Table 5.

# 5   Summary & Conclusion

The goal of this work was to produce a clinically useful technique for automatically classifying activity in ECG recordings. The problem is made challenging by the intra- and inter-patient differences present in the morphology and timing characteristics of the ECG produced by compromised cardiovascular systems and by the variability in the classification tasks that a clinician might want to perform. We propose to address these difficulties with a method for using active learning to perform patient-adaptive and task-adaptive heartbeat classification.

When tested on the most widely used benchmark database of cardiologist annotated ECG recordings, our method had better performance than other recently proposed methods on the two primary classification tasks recommended by AAMI. Additionally, our method required over 90% less training data than the methods to which it was compared. We also showed that our method compares favorably to a state-of-the-art hand coded algorithm for a third common classification task.

To test out the practical applicability of our method, we conducted a small study with two cardiologists. Both cardiologists were able to use our tool with minimal training, and achieved excellent classification results with a small amount of labor per record.

These preliminary results are highly encouraging, and suggest that active learning can be used practically in a clinical setting to not only reduce the labor cost but also garner additional improvements in performance. Of course, there is still room for improvement. In all experiments we used identical input parameters; further tuning of these parameters may improve results. However, in a clinical setting parameter tuning is impractical, and thus more work to investigate automated parameter tuning is needed. Based on preliminary experiments we believe that by first learning the optimal number of initial clusters for each record one can improve performance while decreasing the total number of required labels. It may also be possible to further reduce the amount of required expert labor by starting with a global classifier and then adapting it using active learning.

**Acknowledgments**

We would like to thank Benjamin Scirica, Collin Stultz, and Zeeshan Syed for sharing their expert knowledge in cardiology and for their participation in our experiments. This work was supported in part by the NSERC and by Quanta Computer Inc.

# References

[1] D. V. Exner, K. M. Kavanagh, M. P. Slawnych et al, and for the REFINE Investigators. Noninvasive risk assessment early after a myocardial infarction: The REFINE study. *J Am Coll Cardiol*, 50(24):2275–2284, 2007.

[2] Z. Syed, B. Scirica, S. Mohanavel, P. Sung, C. Cannon, P. Stone, C. Stultz, and J. V. Guttag. Relation to death within 90 days of non-st-elevation acute coronary syndromes to variability in electrocardiographic morphology. *Am J of Cardiol*, 103(3), 2009.

[3] P. de Chazal and R. B. Reilly. A Patient-Adapting Heartbeat Classifier Using ECG Morphology and Heartbeat Interval Features. *Biomedical Engineering, IEEE Transactions on*, 53(12):2535–2543, Dec. 2006.

[4] Y. H. Hu, S. Palreddy, and W.J. Tompkins. A Patient-Adaptable ECG Beat Classifier Using a Mixture of Experts Approach. *Biomedical Engineering, IEEE Transactions on*, 44(9):891–900, Sept. 1997.

[5] T. Ince, S. Kiranyaz, and M. Gabbouj. A generic and robust system for automated patient-specific classification of ecg signals. *IEEE Transactions on Biomedical Engineering*, 56(5), May 2009.

[6] P. Hamilton. Open Source ECG Analysis. In *Computers in Cardiology*, volume 29, pages 101–104, 2002.

[7] J. Bigger, F. Dresdale, and R. Heissenbuttel et. al. Ventricular arrhythmias in ischemic heart disease: mechanism, prevalence, significance, and management. *Prog Cardiovasc Dis*, 19:255, 1977.

[8] T. Smilde, D. van Veldhuisen, and M. van den Berg. Prognostic value of heart rate variability and ventricular arrhythmias during 13-year follow up in patients with mild to moderate heart failure. *Clinical Research in Cardiology*, 98(4):233–239, 2009.

[9] A. L. Goldberger, L. A. N. Amaral, and L. Glass et al. PhysioBank, PhysioToolkit, and PhysioNet: Components of a new research resource for complex physiologic signals. *Circulation*, 101(23):e215–e220, 2000 (June 13). Circulation Electronic Pages: http://circ.ahajournals.org/cgi/content/full/101/23/e215.

[10] P. de Chazal, M. O'Dwyer, R. B. Reilly, and Senior Member. Automatic Classification of Heartbeats Using ECG Morphology and Heartbeat Interval Features. *IEEE Transactions on Biomedical Engineering*, 51:1196–1206, 2004.

[11] K. Sternickel. Automatic pattern recognition in ecg time series. In *Computer Methods and Programs in Biomedicine, Vol: 68*, pages 109–115, 2002.

[12] Z. Syed, J. Guttag, and C. Stultz. Clustering and Symbolic Analysis of Cardiovascular Signals: Discovery and Visualization of Medically Relevant Patterns in Long-term Data Using Limited Prior Knowledge. *EURASIP Journal on Advances in Signal Processing*, 2007:97–112, 2007.

[13] S. Tong and D. Koller. Support vector machine active learning with applications to text classification. *Journal of Machine Learning Research*, 2:45–66, 2002.

[14] S. Dasgupta and D. Hsu. Hierarchical sampling for active learning. In *ICML '08: Proceedings of the 25th international conference on Machine learning*, pages 208–215, New York, NY, USA, 2008. ACM.

[15] Z. Xu, K. Yu, V. Tresp, X. Xu, and J. Wang. Representative sampling for text classification using support vector machines. In *Proceedings of the twenty-fifth European Conference on Information Retrieval*, pages 393–407. Springer, 2003.

[16] H.T. Nguyen and A. Smeulders. Active learning using pre-clustering. In *Proceedings of the twenty-first international conference on Machine learning*, page 79, New York, NY, USA, 2004. ACM.

[17] J. H. Ward. Hierarchical grouping to optimize an objective function. *Journal of the American Statistical Association*, 58(301):234–244, 1963.

[18] S. Kamvar, D. Klein, and C. Manning. Interpreting and extending classical agglomerative clustering algorithms using a model-based approach. In *Proceedings of nineteenth International Conference on Machine Learning*, pages 283–290, 2002.

[19] T. Joachims. *Making Large-scale Support Vector Machine Learning Practical*. MIT Press, Cambridge, MA, USA, 1999.

[20] M. Sokolova, N. Japkowicz, and S. Szpakowicz. *Beyond Accuracy, F-score and ROC: a Family of Discriminant Measures for Performance Evaluation*, volume 4304 of *Lecture Notes in Computer Science*, pages 1015–1021. Springer Berlin/Heidelberg, 2006.

